# Intersecting regions: The key to combinatorial structure in hidden unit space

**Janet Wiles**
Depts of Psychology and
Computer Science,
University of Queensland
QLD 4072 Australia.
janetw@cs.uq.oz.au

**Mark Ollila,**
Vision Lab, CITRI
Dept of Computer Science,
University of Melbourne,
Vic 3052 Australia
molly@vis.citri.edu.au

## Abstract

Hidden units in multi-layer networks form a representation space in which each region can be identified with a class of equivalent outputs (Elman, 1989) or a logical state in a finite state machine (Cleeremans, Servan-Schreiber & McClelland, 1989; Giles, Sun, Chen, Lee, & Chen, 1990). We extend the analysis of the spatial structure of hidden unit space to a combinatorial task, based on binding features together in a visual scene. The logical structure requires a combinatorial number of states to represent all valid scenes. On analysing our networks, we find that the high dimensionality of hidden unit space is exploited by using the intersection of neighboring regions to represent conjunctions of features. These results show how combinatorial structure can be based on the spatial nature of networks, and not just on their emulation of logical structure.

## 1 TECHNIQUES FOR ANALYSING THE SPATIAL AND LOGICAL STRUCTURE OF HIDDEN UNIT SPACE

In multi-layer networks, regions of hidden unit space can be identified with classes of equivalent outputs. For example, Elman (1989) showed that the hidden unit patterns for words in simple grammatical sentences cluster into regions, with similar patterns representing similar grammatical entities. For example, different tokens of the same word are clustered tightly, indicating that they are represented within a small region. These regions can be grouped into larger regions, reflecting a hierarchical structure. The largest

groups represent the abstract categories, nouns and verbs. Elman used cluster analysis to demonstrate this hierarchical grouping, and principal component analysis (*PCA*) to show dimensions of variation in the representation in hidden unit space.

An alternative approach to Elman's hierarchical clustering is to identify each region with a functional state. By tracing the trajectories of sequences through the different regions, an equivalent finite state machine (*FSM*) can be constructed. This approach has been described using Reber grammars with simple recurrent networks (Cleeremans, Servan-Schreiber & McClelland, 1989) and higher-order networks (Giles, Sun, Chen, Lee, & Chen, 1990). Giles et al. showed that the logical structure of the grammars is embedded in hidden unit space by identifying each regions with a state, extracting the equivalent finite state machine from the set of states, and then reducing it to the minimal FSM.

Clustering and FSM extraction demonstrate different aspects of representations in hidden unit space. Elman showed that regions can be grouped hierarchically and that dimensions of variation can be identified using PCA, emphasizing how the functionality is reflected in the spatial structure. Giles et al. extracted the logical structure of the finite state machine in a way that represented the logical states independently of their spatial embedding. There is an inherent trade off between the spatial and logical analyses: In one sense, the FSM is the idealized version of a grammar, and indeed for the Reber grammars, Giles et al. found improved performance on the extracted FSMs over the trained networks. However, the states of the FSM increase combinatorially with the size of the input. If there is information encoded in the hierarchical grouping of regions or relative spatial arrangement of clusters, the extracted FSM cannot exploit it.

The basis of the logical equivalence of a FSM and the hidden unit representations is that disjoint regions of hidden unit space represent separate logical states. In previous work, we reversed the process of identifying clusters with states of a FSM, by using prior knowledge of the minimal FSM to label hidden unit patterns from a network trained on sequences from three temporal functions (Wiles & Bloesch, 1992). Canonical discriminant analysis (*CDA*, Cliff, 1987) was then used to view the hidden unit patterns clustered into regions that corresponded to the six states of the minimal FSM.

In this paper we explore an alternative interpretation of regions. Instead of considering disjoint regions, we view each region as a sub-component lying at the *intersection* of two or more larger regions. For example, in the three-function simulations, the six clusters can be interpreted in terms of three large regions that identify the three possible temporal functions, overlapping with two large regions that identify the output of the network (see Figure 1). The six states can then be seen as combinations of the three function and two output classes (i.e, 5 large overlapping regions instead of 6 smaller disjoint ones). While the three-function simulation does provide a clear demonstration of the intersecting structure of regions, nonetheless, only six states are required to represent the minimal FSM and harder tasks are needed to demonstrate combinatorial representations.

## 2  SIMULATIONS OF THE CONJUNCTION OF COLOR, SHAPE AND LOCATION

The representation of combinatorial structure is an important aspect of any computational task because of the drastic implications of combinatorial explosion for scaling. The intersection of regions is a concise way to represent all possible combinations of different items. We demonstrate this idea applied to the analysis of a hidden unit space

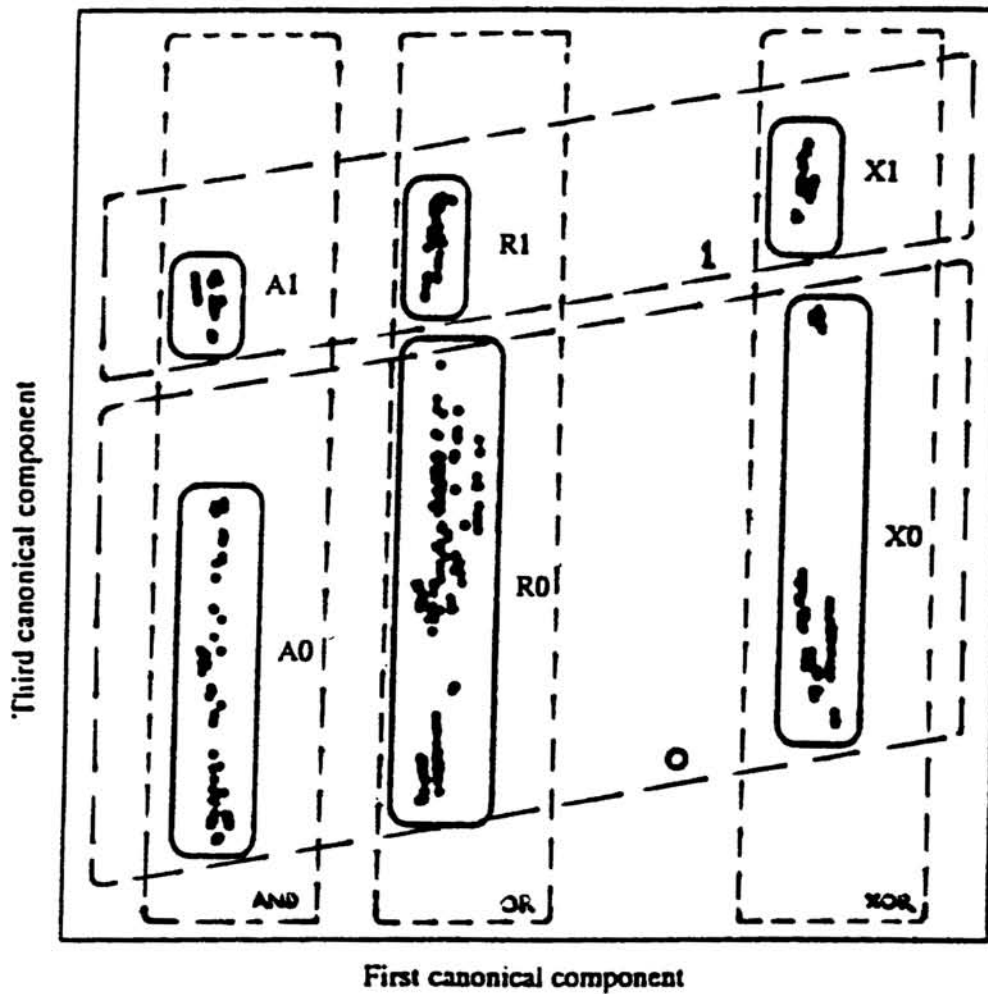

*Figure 1. Intersecting regions in hidden unit space.* Hidden unit patterns from the three-function task of Wiles and Bloesch (1992) are shown projected onto the first and third canonical components. Each temporal function, XOR, AND and OR is represented by a vertical region, separated along the first canonical component. The possible outputs, 0 and 1 are represented by horizontal regions, separated down the third canonical component. The states of the finite state machine are represented by the regions in the intersections of the vertical and horizontal regions. (Adapted from Wiles & Bloesch, 1992, Figure 1b.)

representation of conjunctions of colors, shapes and locations. In our task, a *scene* consists of zero or more objects, each object identified by its color, shape and location. The number of scenes, $C$, is given by $C = (sf+1)^l$ where $s, f$, and $l$ are the numbers of shapes, features and locations respectively. This problem illustrates several important components: There is no unique representation of an object in the input or output – each object is represented only by the presence of a shape and color at a given location. The task of the network is to create hidden unit representations for all possible scenes, each containing the features themselves, and the binding of features to position.

The simulations involved two locations, three possible shapes and three colors (100 legitimate scenes). A 12–20–12 encoder network was trained on the entire set of scenes and the hidden unit patterns for each scene were recorded. Analysis using CDA with 10 groups designating all possible combinations of zero, one or two colors showed that the hidden unit space was partitioned into intersecting regions corresponding to the three colors or no color (see Figure 2a). CDA was repeated using groups designating all combinations of shapes, which showed an alternative partitioning into four intersecting regions related to the component shapes (see Figure 2b). Figures 2a and 2b show alternate two-dimensional projections of the 20-dimensional space. The analyses showed that each hidden unit pattern was contained in many different groupings, such as all objects that are red, all triangles, or all red triangles. In linguistic terms, each hidden unit pattern corresponds to a *token* of a feature, and the region containing all tokens of a given group corresponds to its abstract *type*. The interesting aspect of this representation is that the network had learnt not only how to separate the groups, but also to use overlapping regions. Thus given a region that represents a circle and one representing a triangle, the intersection of the two regions implies a scene that has both a circle and a triangle.

Given suitable groups, the perspectives provided by CDA show many different abstract types within the hidden unit space. For example, scenes can be grouped according to the number of objects in a scene, or the number of squares in a scene. We were initially surprised that contiguous regions exist for representing scenes with zero, one and two objects, since the output units only require representations of individual features, such as square or circle, and not the abstraction to "any shape", or even more abstract, "any object". It seems plausible that the separation of these regions is due to the high dimensionality provided by 12–20–12 mappings. The excess degrees of freedom in hidden unit space can encode variation in the inputs that is not necessarily required to complete the task. With fewer hidden units, we would expect that variation in the input patterns that is not required for completing the task would be compressed or lost under the competing requirement of maximally separating functionally useful groups in the hidden unit space. This explanation found support in a second simulation, using a 12–8–12 encoder network. Whereas analysis of the 12–20–12 network showed separation of patterns into disjoint regions by number of objects, the smaller 12–8–12 network did not. Over all, our analyses showed that as the number of dimensions increases, additional aspects of scenes may be represented, even if those aspects are not required for the task that the network is learning.

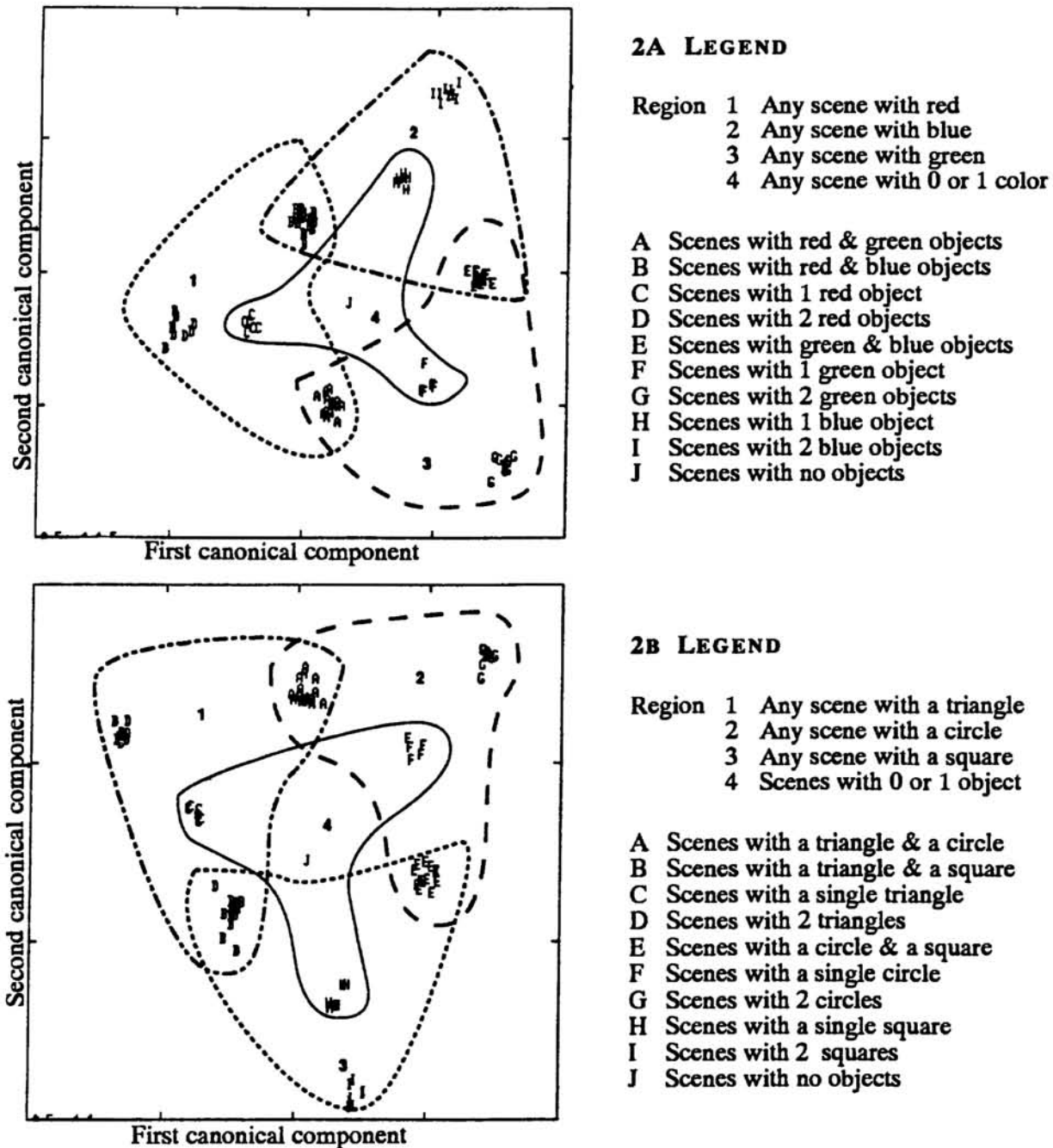

**2A LEGEND**

Region 1  Any scene with red
       2  Any scene with blue
       3  Any scene with green
       4  Any scene with 0 or 1 color

A  Scenes with red & green objects
B  Scenes with red & blue objects
C  Scenes with 1 red object
D  Scenes with 2 red objects
E  Scenes with green & blue objects
F  Scenes with 1 green object
G  Scenes with 2 green objects
H  Scenes with 1 blue object
I  Scenes with 2 blue objects
J  Scenes with no objects

**2B LEGEND**

Region 1  Any scene with a triangle
       2  Any scene with a circle
       3  Any scene with a square
       4  Scenes with 0 or 1 object

A  Scenes with a triangle & a circle
B  Scenes with a triangle & a square
C  Scenes with a single triangle
D  Scenes with 2 triangles
E  Scenes with a circle & a square
F  Scenes with a single circle
G  Scenes with 2 circles
H  Scenes with a single square
I  Scenes with 2 squares
J  Scenes with no objects

*Figure 2. CDA plots showing the representations of features in a scene.* A scene consists of zero, one or two objects, represented in terms of color, shape and location. *2a. Patterns labelled by color:* Hidden unit patterns form ten distinct clusters, which have been grouped into four intersecting regions, 1-4. For example, the hidden unit patterns within region 1 all contain at least one red object, those in regions 2 contain at least one blue one, and those in the intersection of regions 1 and 2 contain one red and one blue object. *2b. Patterns labelled by shape:* Again the hidden unit patterns form ten distinct clusters, which have been grouped into four intersecting regions, however, these regions represent scenes with the same shape. 2a and 2b show alternate groupings of the same hidden unit space, projected onto different canonical components. The two projections can be combined in the mind's eye (albeit with some difficulty) to form a four dimensional representation of the spatial structure of intersecting regions of both color and shape.

# 3 THE SPATIAL STRUCTURE OF HIDDEN UNIT SPACE IS ISOMORPHIC TO THE COMBINATORIAL STRUCTURE OF THE VISUAL MAPPING TASK

In conclusion, the simulations demonstrate how combinatorial structure can be embedded in the spatial nature of networks in a way that is isomorphic to the combinatorial structure of the task, rather than by emulation of logical structure. In our approach, the representation of intersecting regions is the key to providing combinatorial representations. If the visual mapping task were extended by including a feature specifying the color of the background scene (e.g., blue or green) the number of possible scenes would double, as would the number of states in a FSM. By contrast, in the hidden unit representation, the additional feature would involve adding two more overlapping regions to those currently supported by the spatial structure. This could be implemented by dividing hidden unit space along an unused dimension, orthogonal to the current groups.

The task presented in this case study is extremely simplified, in order to expose the intrinsic combinatorial structure required in binding. Despite the simplifications, it does contain elements of tasks that face real cognitive systems. In the simulations above, individual objects can be clustered by their shape or color, or whole scenes by other properties, such as the number of squares in the scene. These representations provide a concise and easily accessible structure that solves the combinatorial problem of binding several features to one object, in such a way as to represent the individual object, and yet also allow efficient access to its component features. The flexibility of such access processes is one of the main motivations for tensor models of human memory (Humphreys, Bain & Pike, 1989) and analogical reasoning (Halford et al., in press). Our analysis of spatial structure in terms of intersecting regions has a straightforward interpretation in terms of tensors, and provides a basis for future work on network implementations of the tensor memory and analogical reasoning models.

## Acknowledgements

We thank Simon Dennis and Steven Phillips for their canonical discriminant program. This work was supported by grants from the Australian Research Council.

## References

Cleeremans, A., Servan-Schreiber, D., and McClelland, J.L. (1989). Finite state automata and simple recurrent networks, *Neural Computation*, 1, 372-381.

Cliff, N. (1987). *Analyzing Multivariate Data*. Harcourt Brace Jovanovich, Orlando, Florida.

Elman, J. (1989). Representation and structure in connectionist models. CRL Technical Report 8903, Center for Research in Language, University of California, San Diego, 26pp.

Giles, C. L., Sun, G. Z., Chen, H. H., Lee, Y. C., and Chen, D. (1990). Higher Order Recurrent Networks. In D.S. Touretzky (ed.) *Advances in Neural Information Processing Systems 2*, Morgan-Kaufmann, San Mateo, Ca., 380-387.

Halford, G.S., Wilson, W.H., Guo, J., Wiles, J. and Stewart, J.E.M. Connectionist implications for processing capacity limitations in analogies. To appear in K.J. Holyoak & J. Barnden (Eds.), *Advances in Connectionist and Neural Computation Theory, Vol 2: Analogical Connections*. Norwood, NJ: Ablex, in press.

Humphreys, M.S., Bain, J.D., and Pike, R. (1989). Different ways to cue a coherent memory system: A theory of episodic, semantic and procedural tasks, *Psychological Review, 96* (2), 208-233.

Wiles, J. and Bloesch, A. (1992). Operators and curried functions: Training and analysis of simple recurrent networks. In J. E. Moody, S. J. Hanson, and R. P. Lippmann (Eds.) *Advances in Neural Information Processing Systems 4*, Morgan-Kaufmann, San Mateo, Ca.
